# A Comparative Study Of A Modified Bumptree Neural Network With Radial Basis Function Networks and the Standard Multi-Layer Perceptron.

**Richard T.J. Bostock and Alan J. Harget**
Department of Computer Science & Applied Mathematics
Aston University
Birmingham
England

## Abstract

Bumptrees are geometric data structures introduced by Omohundro (1991) to provide efficient access to a collection of functions on a Euclidean space of interest. We describe a modified bumptree structure that has been employed as a neural network classifier, and compare its performance on several classification tasks against that of radial basis function networks and the standard mutli-layer perceptron.

## 1 INTRODUCTION

A number of neural network studies have demonstrated the utility of the multi-layer perceptron (MLP) and shown it to be a highly effective paradigm. Studies have also shown, however, that the MLP is not without its problems, in particular it requires an extensive training time, is susceptible to local minima problems and its performance is dependent upon its internal network architecture. In an attempt to improve upon the generalisation performance and computational efficiency a number of studies have been undertaken principally concerned with investigating the parametrisation of the MLP. It is well known, for example, that the generalisation performance of the MLP is affected by the number of hidden units in the network, which have to be determined empirically since theory provides no guidance. A number of investigations have been conducted into the possibility of automatically determining the number of hidden units during the training phase (Bostock, 1992). The results show that architectures can be attained which give satisfactory, although generally sub-optimal, performance.

Alternative network architectures such as the Radial Basis Function (RBF) network have also been studied in an attempt to improve upon the performance of the MLP network. The RBF network uses basis functions in which the weights are effective over only a small portion of the input space. This is in contrast to the MLP network where the weights are used in a more global fashion, thereby encoding the characteristics of the training set in a more compact form. RBF networks can be rapidly trained thus making

them particularly suitable for situations where on-line incremental learning is required. The RBF network has been successfully applied in a number of areas such as speech recognition (Renals, 1992) and financial forecasting (Lowe, 1991). Studies indicate that the RBF network provides a viable alternative to the MLP approach and thus offers encouragement that networks employing local solutions are worthy of further investigation.

In the past few years there has been an increasing interest in neural network architectures based on tree structures. Important work in this area has been carried out by Omohundro (1991) and Gentric and Withagen (1993). These studies seem to suggest that neural networks employing a tree based structure should offer the same benefits of reduced training time as that offered by the RBF network. The particular tree based architecture examined in this study is the bumptree which provides efficient access to collections of functions on a Euclidean space of interest. A bumptree can be viewed as a natural generalisation of several other geometric data structures including oct-trees, k-d trees, balltrees (Omohundro, 1987) and boxtrees (Omohundro, 1989).

In this paper we present the results of a comparative study of the performance of the three types of neural networks described above over a wide range of classification problems. The performance of the networks was assessed in terms of the percentage of correct classifications on a test, or generalisation data set, and the time taken to train the network. Before discussing the results obtained we shall give an outline of the implementation of our bumptree neural network since this is more novel than the other two networks.

## 2   THE BUMPTREE NEURAL NETWORK

Bumptree neural networks share many of the underlying principles of decision trees but differ from them in the manner in which patterns are classified. Decision trees partition the problem space into increasingly small areas. Classification is then achieved by determining the lowest branch of the tree which contains a reference to the specified point. The bumptree neural network described in this paper also employs a tree based structure to partition the problem space, with each branch of the tree being based on multiple dimensions. Once the problem space has been partitioned then each branch can be viewed as an individual neural network modelling its own local area of the problem space, and being able to deal with patterns from multiple output classes.

Bumptrees model the problem space by subdividing the space allowing each division to be described by a separate function. Initial partitioning of the problem space is achieved by randomly assigning values to the root level functions. A learning algorithm is applied to determine the area of influence of each function and an associated error calculated. If the error exceeds some threshold of acceptability then the area in question is further subdivided by the addition of two functions; this process continues until satisfactory performance is achieved. The bumptree employed in this study is essentially a binary tree in which each leaf of the tree corresponds to a function of interest although the possibility exists that one of the functions could effectively be redundant if it fails to attract any of the patterns from its parent function.

A number of problems had to be resolved in the design and implementation of the bumptree. Firstly, an appropriate procedure had to be adopted for partitioning the

problem space. Secondly, consideration had to be given to the type of learning algorithm to be employed. And finally, the mechanism for calculating the output of the network had to be determined. A detailed discussion of these issues and the solutions adopted now follows.

## 2.1     PARTITIONING THE PROBLEM SPACE

The bumptree used in this study employed gaussian functions to partition the problem space, with two functions being added each time the space was partitioned. Patterns were assigned to whichever of the functions had the higher activation level with the restriction that the functions below the root level could only be active on patterns that activated their parents. To calculate the activation of the gaussian function the following expression was used:

$$A_{fp} = \exp^{-\left(0.5*((In_{pi}*C_{fi})/a_{fi})\right)^2} * 1/(a_{fi}*\sqrt{3.14159*2}) \qquad (1)$$

where $A_{fp}$ is the activation of function f on pattern p over all the input dimensions, $a_{fi}$ is the radius of function f in input dimension i, $C_{fi}$ is the centre of function f in input dimension i, and $In_{pi}$ is the ith dimension of the pth input vector.

It was found that the locations and radii of the functions had an important impact on the performance of the network. In the original bumptree introduced by Omohundro every function below the root level was required to be wholly enclosed by its parent function. This restriction was found to degrade the performance of the bumptree particularly if a function had a very small radius since this would produce very low levels of activation for most patterns. In our studies we relaxed this constraint by assigning the radius of each function to one, since the data presented to the bumptree was always normalised between zero and one. This modification led to an improved performance.

A number of different techniques were examined in order to effectively position the functions in the problem space. The first approach considered, and the simplest, involved selecting two initial sets of centres for the root function with the centre in each dimension being allocated a value between zero and one. The functions at the lower levels of the tree were assigned in a similar manner with the requirement that their centres fell within the area of the problem space for which their parent function was active. The use of non-hierarchical clustering techniques such as the Forgy method or the K-means clustering technique developed by MacQueen provided other alternatives for positioning the functions. The approach finally adopted for this study was the multiple-initial function (MIF) technique.

In the MIF procedure ten sets of functions centres were initially defined by random assignment and each pattern in the training set assigned to the function with the highest activation level. A "goodness" measure was then determined for each function over all patterns for which the function was active. The goodness measure was defined as the square of the error between the calculated and observed values divided by the number of active patterns. The function with the best value was retained and the remaining functions that were active on one or more patterns had their centres averaged in each dimension to provide a second function. The functions were then added to the network structure and the patterns assigned to the function which gave the greater activation.

## 2.2   THE LEARNING ALGORITHM

A bumptree neural network comprises a number of functions each function having its own individual weight and bias parameters and each function being responsive to different characteristics in the training set. The bumptree employed a weighted value for every input to output connection and a single bias value for each output unit. Several different learning algorithms for determining the weight and bias values were considered together with a genetic algorithm approach (Williams, 1993). A one-shot learning algorithm was finally adopted since this gave good results and was computationally efficient. The algorithm used a pseudo-matrix inversion technique to determine the weight and bias parameters of each function after a single presentation of the relevant patterns in the training set had been made. The output of any function for a given pattern p was determined from

$$ao_{ipz} = \sum_{j=1}^{jmax} \alpha_{ijz} * x_j^{(p)} + \beta_{iz} \tag{2}$$

where $ao_{ipz}$ is the output of the zth output unit of the ith function on the pth pattern, j is the input unit, jmax is the total number of input units, $\alpha_{ijz}$ is the weight that connects the jth input unit to the zth output unit for the ith function, $x_j^{(p)}$ is the element of the pth pattern concerned with the jth input dimension, and $\beta_{iz}$ is the bias value for the zth output unit.

The weight and bias parameters were determined by minimising the squared error given in (3), where $E_i$ is the error of the ith function across all output dimensions (zmax), for all patterns upon which the function is active (pmax). The desired output for the zth output dimension is $tv_{pz}$,, and $\alpha_{oipz}$ is the actual output of the ith function on the zth dimension of the pth pattern. The weight values are again represented by $\alpha_{ijz}$ and the bias by $\beta_{iz}$.

$$E_i = \frac{1}{2} \sum_{p=1}^{pmax} \sum_{z=1}^{zmax} \{ \sum_{j=1}^{jmax} \alpha_{ijz} x_j^{(p)} + \beta_{iz} - tv_{pz} \}^2 \tag{3}$$

After the derivatives of $\alpha_{ijz}$ and $\beta_{iz}$ were determined it was a simple task to arrive at the three matrices used to calculate the weight and bias values for the individual functions. Problems were encountered in the matrix inversion when dealing with functions which were only active on a few patterns and which were far removed from the root level of the tree; this led to difficulties with singular matrices. It was found that the problem could be overcome by using the Gauss-Jordan singular decomposition technique for the pseudo-inversion of the matrices.

## 2.3   CALCULATION OF THE NETWORK OUTPUT

The difficulty in determining the output of the bumptree was that there were usually functions at different levels of the tree that gave slightly different outputs for each active pattern. Several different approaches were studied in order to resolve the difficulty including using the normalised output of all the active functions in the tree irrespective of their level in the structure. A technique which gave good results and was used in this

study calculated the output for a pattern solely on the output of the lowest level active function in the tree. The final output class of a pattern being given by the output unit with the highest level of activation.

## 3   NETWORK PERFORMANCES

The performance of the bumptree neural network was compared against that of the standard MLP and RBF networks on a number of different problems. The bumptree used the MIF placing technique in which the radius of each function was set to one. This particular implementation of the bumptree will now be referred to as the MIF bumptree. The MLP used the standard backpropagation algorithm (Rumelhart, 1986) with a learning rate of 0.25 and a momentum value of 0.9. The initial weights and bias values of the network were set to random values between -2 and +2. The number of hidden units assigned to the network was determined empirically over several runs by varying the number of hidden units until the best generalisation performance was attained. The RBF network used four different types of function, they were gaussian, multi-quadratic, inverse multi-quadratic and thin plate splines. The RBF network placed the functions using sample points within the problem space covered by the training set.

### 3.1   INITIAL STUDIES

In the initial studies, a set of classical non-linear problems was used to compare the performance of the three types of networks. The set consisted of the XOR, Parity(6) and Encoder(8) problems. The average results obtained over 10 runs for each of the data sets are shown in Table 1 - the figures presented are the percentage of patterns correctly classified in the training set together with the standard deviation.

Table 1. Percentage of Patterns Correctly Classified for the three Data Sets for each Network type.

| DATA SET | MLP | RBF | MIF |
|---|---|---|---|
| XOR | 100 | 100 | 100 |
| Parity(6) | 100 | 92.1 ± 4.7 | 98.3 ± 4.2 |
| Encoder(8) | 100 | 82.5 ± 16.8 | 100 |

For the XOR problem the MLP network required an average of 222 iterations with an architecture of 4 hidden units, for the parity problem an architecture of 10 hidden units and an average of 1133 iterations, and finally for the encoder problem the network required an average of 1900 iterations for an architecture consisting of three hidden units.

The RBF network correctly classified all the patterns of the XOR data set when four multi-quadratic, inverse multi-quadratic or gaussian functions were used. For the parity(6) problem the best result was achieved with a network employing between 60 and 64 inverse multi-quadratic functions. In the case of the encoder problem the best performance was obtained using a network of 8 multi-quadratic functions.

The MIF bumptree required two functions to achieve perfect classification for the XOR and encoder problems and an average of 40 functions in order to achieve the best performance on the parity problem. Thus in the case of the XOR and encoder problems no further functions were required additional to the root functions.

A comparison of the training times taken by each of the networks revealed considerable differences. The MLP required the most extensive training time since it used the backpropagation training algorithm which is an iterative procedure. The RBF network required less training time than the MLP, but suffered from the fact that for all the patterns in the training set the activity of all the functions had to be calculated in order to arrive at the optimal weights. The bumptree proved to have the quickest training time for the parity and encoder problems and a training time comparable to that taken by the RBF network for the XOR problem. This superiority arose because the bumptree used a non-iterative training procedure, and a function was only trained on those members of the training set for which the function was active.

In considering the sensitivity of the different networks to the parameters chosen some interesting results emerge. The performance of the MLP was found to be dependent on the number of hidden units assigned to the network. When insufficient hidden units were allocated the performance of the MLP degraded. The performance of the RBF network was also found to be highly influenced by the values taken for various parameters, in particular the number and type of functions employed by the network. The bumptree on the other hand was assigned the same set of parameters for all the problems studied and was found to be less sensitive than the other two networks to the parameter settings.

## 3.2    COMPARISON OF GENERALISATION PERFORMANCE

The performance of the three different networks was also measured for a set of four 'real-world' problems which allowed the generalisation performance of each network to be determined. A summary of the results taken over 10 runs is given in Table 2.

Table 2 Performance of the Networks on the Training and Generalisation Data Sets of the Test Problems.

| DATA | NETWORK | FUNCTIONS HIDDEN UNITS | TRAINING | TEST |
|---|---|---|---|---|
| Iris | | | | |
| | MLP | 4 | 100 | 95.7 ± 0.6 |
| | RBF | 75 gaussians | 100 | 96.0 ± 0.0 |
| | MIF | 8 | 100 | 97.5 ± 0.4 |
| Skin Cancer | | | | |
| | MLP | 6 | 88.7 ± 4.3 | 79.2 ± 1.7 |
| | RBF | 10 multi-quad | 84.4 ± 3.2 | 80.3 ± 4.4 |
| | MIF | 4 | 79.8 ± 5.2 | 80.8 ± 1.9 |
| Vowel Data | | | | |
| | MLP | 20 | 82.4 ± 5.3 | 77.1 ± 6.6 |
| | RBF | 50 Thin plate spl. | 82.1 ± 1.5 | 77.8 ± 1.4 |
| | MIF | 104 | 86.5 ± 5.6 | 73.6 ± 4.6 |
| Diabetes | | | | |
| | MLP | 16 | 82.5 ± 2.7 | 78.9 ± 1.2 |
| | RBF | 25 Thin plate spl. | 76.0 ± 0.8 | 78.9 ± 0.9 |
| | MIF | 3 | 76.5 ± 1.2 | 80.0 ± 1.1 |

All three networks produce a comparable performance on the test problems, but in the case of the bumptree this was achieved with a training time substantially less than that required by the other networks. Inspection of the results also shows that the bumptree required fewer functions in general than the RBF network.

The results shown above for the bumptree were obtained with the same set of parameters used in the initial study which further confirms its lack of sensitivity to parameter settings.

## 4.    CONCLUSION

A comparative study of the performance of three different types of networks, one of which is novel, has been conducted on a wide range of problems. The results show that the performance of the bumptree compared very favourably, both in terms of generalisation and training times, with the more traditional MLP and RBF networks. In addition, the performance of the bumptree proved to be less sensitive to the parameters settings than the other networks. These results encourage us to continue further investigation of the bumptree neural network and lead us to conclude that it has a valid place in the list of current neural networks.

## Acknowledgement
We gratefully acknowledge the assistance given by Richard Rohwer.

## References
Bostock R.T.J. & Harget A.J. (1992) Towards a Neural Network Based System for Skin Cancer Diagnosis: *IEE Third International Conference on Artificial Neural Networks*: P215-220.
Broomhead D.S. & Lowe D. (1988) Radial Basis Functions, Multi-Variable Functional Interpolation and Adaptive Networks: *RSRE Memorandum* No. 4148, Royal Signals and Radar Establishment, Malvern, England.
Gentric P. & Withagen H.C.A.M. (1993) Constructive Methods for a New Classifier Based on a Radial Basis Function Network Accelerated by a Tree: *Report, Eindhoven Technical University*, Eindhoven, Holland.
Lowe D. & Webb A.R. (1991) Time Series Prediction by Adaptive Networks: A Dynamical Systems Perspective: *IEE Proceedings-F*, vol. 128(1), Feb.,, P17-24.
Moody J. & Darken C. (1988) Learning With Localized Receptive Fields: *Research Report YALEU/DCS/RR-649*.
Omohundro S.M. (1987) Efficient Algorithms With Neural Network Behaviour; in *Complex Systems* 1 (1987): P273-347.
Omohundro S.M. (1989) Five Balltree Construction Algorithms: *International Computer Science Institute Technical Report* TR-89-063.
Omohundro S.M. (1991) Bumptrees for Efficient Function, Constraint, and Classification Learning: *Advances in Neural Information Processing Systems* 3, P693-699 .
Renals S. & Rohwer R.J. (1989) Phoneme Classification Experiments Using Radial Basis Functions: *Proceedings of the IJCNN*, P461-467.
Rumelhart D.E., Hinton G.E. & Williams R.J. (1986) Learning Internal Representations by Error Propagation: in *Parallel Distributed Processing*, vol.1 P318-362. Cambridge, MA : MIT Press.
Williams B.V., Bostock R.T.J., Bounds D.G. & Harget A.J. (1993) The Genetic Bumptree Classifier: *Proceedings of the BNSS Symposium on Artificial Neural Networks: to be published*.